# Nash Equilibria of Static Prediction Games

**Michael Brückner**
Department of Computer Science
University of Potsdam, Germany
mibrueck@cs.uni-potsdam.de

**Tobias Scheffer**
Department of Computer Science
University of Potsdam, Germany
scheffer@cs.uni-potsdam.de

## Abstract

The standard assumption of identically distributed training and test data is violated when an adversary can exercise some control over the generation of the test data. In a prediction game, a learner produces a predictive model while an adversary may alter the distribution of input data. We study single-shot prediction games in which the cost functions of learner and adversary are not necessarily antagonistic. We identify conditions under which the prediction game has a unique Nash equilibrium, and derive algorithms that will find the equilibrial prediction models. In a case study, we explore properties of Nash-equilibrial prediction models for email spam filtering empirically.

## 1 Introduction

The assumption that training and test data are governed by identical distributions underlies many popular learning mechanisms. In a variety of applications, however, data at application time are generated by an adversary whose interests are in conflict with those of the learner. In computer and network security, fraud detection, and drug design, the distribution of data is changed – by a malevolent individual or under selective pressure – in response to the predictive model.

An adversarial interaction between learner and data generator can be modeled as a single-shot game in which one player controls the predictive model whereas the other player exercises some control over the distribution of the input data. The optimal action for either player generally depends on both players' moves.

The *minimax strategy* minimizes the costs under the worst possible move of the opponent. This strategy is motivated for an opponent whose goal is to inflict the highest possible costs on the learner; it can also be applied when no information about the interests of the adversary is available. Lanckriet et al. [1] study the so called Minimax Probability Machine. This classifier minimizes the maximal probability of misclassifying new instances for a given mean and covariance matrix of each class. El Ghaoui et al. [2] study a minimax model for input data that are known to lie within some hyper-rectangle. Their solution minimizes the worst-case loss over all possible choices of the data in these intervals. Similarly, minimax solutions to classification games in which the adversary deletes input features or performs a feature transformation have been studied [3, 4, 5]. These studies show that the minimax solution outperforms a learner that naively minimizes the costs on the training data without taking the adversary into account.

When rational opponents aim at minimizing their personal costs, then the minimax solution is overly pessimistic. A *Nash equilibrium* is a pair of actions chosen such that no player gains a benefit by unilaterally selecting a different action. If a game has a unique Nash equilibrium, it is the strongest available concept of an optimal strategy in a game against a rational opponent. If, however, multiple equilibria exist and the players choose their action according to distinct ones, then the resulting combination may be arbitrarily disadvantageous for either player. It is therefore interesting to study whether adversarial prediction games have a unique Nash equilibrium.

We study games in which both players – learner and adversary – have cost functions that consist of data-dependent loss and regularizer. Contrasting prior results, we do not assume that the players' cost functions are antagonistic. As an example, consider that a spam filter may minimize the error rate whereas a spam sender may aim at maximizing revenue solicited by spam emails. These criteria are conflicting, but not the exact negatives of each other. We study under which conditions unique Nash equilibria exist and derive algorithms for identifying them.

The rest of this paper is organized as follows. Section 2 introduces the problem setting and defines action spaces and cost functions. We study the existence of a unique Nash equilibrium and derive an algorithm that finds it under defined conditions in Section 3. Section 4 discusses antagonistic loss functions. For this case, we derive an algorithm that finds a unique Nash equilibrium whenever it exists. Section 5 reports on experiments on email spam filtering; Section 6 concludes.

## 2 Modeling the Game

We study prediction games between a *learner* ($v = +1$) and an *adversary* ($v = -1$). We consider *static infinite games*. *Static* or single-shot game means that players make decisions simultaneously; neither player has information about the opponent's decisions. *Infinite* refers to continuous cost functions that leave players with infinitely many strategies to choose from. We constrain the players to select pure (*i.e.,* deterministic) strategies. Mixed strategies and extensive-form games such as Stackelberg, Cournot, Bertrand, and repeated games are not within the scope of this work.

Both players can access an input matrix of training instances $\mathbf{X}$ with outputs $\mathbf{y}$, drawn according to a probability distribution $q(\mathbf{X}, \mathbf{y}) = \prod_{i=1}^{n} q(\mathbf{x}_i, y_i)$. The learner's action $\mathbf{a}_{+1} \in A_{+1}$ now is to choose parameters of a linear model $h_{\mathbf{a}_{+1}}(\mathbf{x}) = \mathbf{a}_{+1}^\mathsf{T}\mathbf{x}$. Simultaneously, the adversary chooses a transformation function $\phi_{\mathbf{a}_{-1}}$ that maps any input matrix $\mathbf{X}$ to an altered matrix $\phi_{\mathbf{a}_{-1}}(\mathbf{X})$. This transformation induces a transition from input distribution $q$ to test distribution $q_{test}$ with $q(\mathbf{X}, \mathbf{y}) = q_{test}(\phi_{\mathbf{a}_{-1}}(\mathbf{X}), \mathbf{y}) = \prod_{i=1}^{n} q_{test}(\phi_{\mathbf{a}_{-1}}(\mathbf{X})_i, y_i)$. Our main result uses a model that implements transformations as matrices $\mathbf{a}_{-1} \in A_{-1} \subseteq \mathbb{R}^{m \times n}$. Transformation $\phi_{\mathbf{a}_{-1}}(\mathbf{X}) = \mathbf{X} + \mathbf{a}_{-1}$ adds perturbation matrix $\mathbf{a}_{-1}$ to input matrix $\mathbf{X}$, *i.e.,* input pattern $\mathbf{x}_i$ is subjected to a perturbation vector $\mathbf{a}_{-1,i}$. If, for instance, inputs are word vectors, the perturbation matrix adds and deletes words.

The possible moves $\mathbf{a} = [\mathbf{a}_{+1}, \mathbf{a}_{-1}]$ constitute the *joint action space* $A = A_{+1} \times A_{-1}$ which is assumed to be nonempty, compact, and convex. Action spaces $A_v$ are parameters of the game. For instance, in spam filtering it is appropriate to constrain $A_{-1}$ such that perturbation matrices contain zero vectors for non-spam messages; this reflects that spammers can only alter spam messages.

Each pair of actions $\mathbf{a}$ incurs costs of $\theta_{+1}(\mathbf{a})$ and $\theta_{-1}(\mathbf{a})$, respectively, for the players. Each player has an individual loss function $\ell_v(y', y)$ where $y'$ is the value of decision function $h_{\mathbf{a}_{+1}}$ and $y$ is the true label. Section 4 will discuss antagonistic loss functions $\ell_{+1} = -\ell_{-1}$. However, our main contribution in Section 3 regards non-antagonistic loss functions. For instance, a learner may minimize the zero-one loss whereas the adversary may focus on the lost revenue.

Both players aim at minimizing their loss over the test distribution $q_{test}$. But, since $q$ and consequently $q_{test}$ are unknown, the cost functions are regularized empirical loss functions over the sample $\phi_{\mathbf{a}_{-1}}(\mathbf{X})$ which reflects test distribution $q_{test}$. Equation 1 defines either player's cost function as player-specific loss plus regularizer. The learner's regularizer $\Omega_{\mathbf{a}_{+1}}$ will typically regularize the capacity of $h_{\mathbf{a}_{+1}}$. Regularizer $\Omega_{\mathbf{a}_{-1}}$ controls the amount of distortion that the adversary may inflict on the data and thereby the extent to which an information payload has to be preserved.

$$\theta_v(\mathbf{a}_v, \mathbf{a}_{-v}) = \sum_{i=1}^{n} \ell_v(h_{\mathbf{a}_{+1}}(\phi_{\mathbf{a}_{-1}}(\mathbf{X})_i), y_i) + \Omega_{\mathbf{a}_v} \tag{1}$$

Each player's cost function depends on the opponent's parameter. In general, there is no value $\mathbf{a}_v$ that maximizes $\theta_v(\mathbf{a}_v, \mathbf{a}_{-v})$ independently of the opponent's choice of $\mathbf{a}_{-v}$. The *minimax solution* $\arg\min_{\mathbf{a}_v} \max_{\mathbf{a}_{-v}} \theta_v(\mathbf{a}_v, \mathbf{a}_{-v})$ minimizes the costs under the worst possible move of the opponent. This solution is optimal for a malicious opponent whose goal is to inflict maximally high costs on the learner. In absence of any information on the opponent's goals, the minimax solution still gives the lowest upper bound on the learner's costs over all possible strategies of the opponent.

If both players – learner and adversary – behave rationally in the sense of minimizing their personal costs, then the *Nash equilibrium* is the strongest available concept of an optimal choice of $\mathbf{a}_v$. A

Nash equilibrium is defined as a pair of actions $\mathbf{a}^* = [\mathbf{a}^*_{+1}, \mathbf{a}^*_{-1}]$ such that no player can benefit from changing the strategy unilaterally. That is, for both players $v \in \{-1, +1\}$,

$$\theta_v(\mathbf{a}^*_v, \mathbf{a}^*_{-v}) = \min_{\mathbf{a}_v \in A_v} \theta_v(\mathbf{a}_v, \mathbf{a}^*_{-v}). \tag{2}$$

The Nash equilibrium has several catches. Firstly, if the adversary behaves irrationally in the sense of inflicting high costs on the other player at the expense of incurring higher personal costs, then choosing an action according to the Nash equilibrium may result in higher costs than the minimax solution. Secondly, a game may not have an equilibrium point. If an equilibrium point exists, the game may thirdly possess multiple equilibria. If $\mathbf{a}^* = [\mathbf{a}^*_{+1}, \mathbf{a}^*_{-1}]$ and $\mathbf{a}' = [\mathbf{a}'_{+1}, \mathbf{a}'_{-1}]$ are distinct equilibria, and each player decides to act according to one of them, then a combination $[\mathbf{a}^*_v, \mathbf{a}'_{-v}]$ may be a poor joint strategy and may give rise to higher costs than a worst-case solution. However, if a unique Nash equilibrium exists and both players seek to minimize their individual costs, then the Nash equilibrium is guaranteed to be the optimal move.

## 3  Solution for Convex Loss Functions

In this section, we study the existence of a unique Nash equilibrium of prediction games with cost functions as in Equation 1. We derive an algorithm that identifies the unique equilibrium if sufficient conditions are met. We consider regularized player-specific loss functions $\ell_v(y', y)$ which are *not* assumed to satisfy the antagonicity criterion $\ell_{+1} = -\ell_{-1}$. Both loss functions are, however, required to be convex and twice differentiable, and we assume strictly convex regularizers $\Omega_{\mathbf{a}_v}$ such as the $l_2$-norm regularizer. Player- and instance-specific costs may be attached to the loss functions; however, we omit such cost factors for greater notational harmony. This section's main result is that if both loss functions are monotonic in $y'$ with different monotonicities – that is, one is monotonically increasing, and one is decreasing for any fixed $y$ – then the game has a unique Nash equilibrium that can be found efficiently.

**Theorem 1.** *Let the cost functions be defined as in Equation 1 with strictly convex regularizers $\Omega_{\mathbf{a}_v}$, let action spaces $A_v$ be nonempty, compact, and convex subsets of finite-dimensional Euclidean spaces. If for any fixed $y$, both loss functions $\ell_v(y', y)$ are monotonic in $y' \in \mathbb{R}$ with distinct monotonicity, convex in $y'$, and twice differentiable in $y'$, then a unique Nash equilibrium exists.*

*Proof.* The players' regularizers $\Omega_{\mathbf{a}_v}$ are strictly convex, and both loss functions $\ell_v(h_{\mathbf{a}_{+1}}(\phi_{\mathbf{a}_{-1}}(\mathbf{X})_i), y_i)$ are convex and twice differentiable in $\mathbf{a}_v \in A_v$ for any fixed $\mathbf{a}_{-v} \in A_{-v}$. Hence, both cost functions $\theta_v$ are continuously differentiable and strictly convex, and according to Theorem 4.3 in [6], at least one Nash equilibrium exists. As each player has an own nonempty, compact, and convex action space $A_v$, Theorem 2 of [7] applies as well; that is, if function

$$\sigma_r(\mathbf{a}) = r\theta_{+1}(\mathbf{a}_{+1}, \mathbf{a}_{-1}) + (1 - r)\theta_{-1}(\mathbf{a}_{+1}, \mathbf{a}_{-1}) \tag{3}$$

is diagonally strictly convex in $\mathbf{a}$ for some fixed $0 < r < 1$, then a unique Nash equilibrium exists. A sufficient condition for $\sigma_r(\mathbf{a})$ to be diagonally strictly convex is that matrix $J_r(\mathbf{a})$ in Equation 4 is positive definite for any $\mathbf{a} \in A$ (see Theorem 6 in [7]). This matrix

$$J_r(\mathbf{a}) = \begin{bmatrix} r\nabla^2_{\mathbf{a}_{+1}\mathbf{a}_{+1}}\theta_{+1}(\mathbf{a}) & r\nabla^2_{\mathbf{a}_{+1}\mathbf{a}_{-1}}\theta_{+1}(\mathbf{a}) \\ (1-r)\nabla^2_{\mathbf{a}_{-1}\mathbf{a}_{+1}}\theta_{-1}(\mathbf{a}) & (1-r)\nabla^2_{\mathbf{a}_{-1}\mathbf{a}_{-1}}\theta_{-1}(\mathbf{a}) \end{bmatrix} \tag{4}$$

is the Jacobian of the pseudo-gradient of $\sigma_r(\mathbf{a})$, that is,

$$g_r(\mathbf{a}) = \begin{bmatrix} r\nabla_{\mathbf{a}_{+1}}\theta_{+1}(\mathbf{a}) \\ (1-r)\nabla_{\mathbf{a}_{-1}}\theta_{-1}(\mathbf{a}) \end{bmatrix}. \tag{5}$$

We want to show that $J_r(\mathbf{a})$ is positive definite for some fixed $r$ if both loss functions $\ell_v(y', y)$ have distinct monotonicity and are convex in $y'$. Let $\ell'_v(y', y)$ be the first and $\ell''_v(y', y)$ be the second derivative of $\ell_v(y', y)$ with respect to $y'$. Let $\mathbf{A}_i$ denote the matrix where the $i$-th column is $\mathbf{a}_{+1}$ and all other elements are zero, let $\mathbf{\Gamma}_v$ be the diagonal matrix with diagonal elements $\gamma_{v,i} = \ell''_v(h_{\mathbf{a}_{+1}}(\phi_{\mathbf{a}_{-1}}(\mathbf{X})_i), y_i)$, and we define $\mu_{v,i} = \ell'_v(h_{\mathbf{a}_{+1}}(\phi_{\mathbf{a}_{-1}}(\mathbf{X})_i), y_i)$. Using these defini-

tions, the Jacobian of Equation 4 can be rewritten,

$$
J_r(\mathbf{a}) = \begin{bmatrix} \phi_{\mathbf{a}_{-1}}(\mathbf{X}) & \mathbf{0} \\ \mathbf{0} & \mathbf{A}_1 \\ \vdots & \vdots \\ \mathbf{0} & \mathbf{A}_n \end{bmatrix} \begin{bmatrix} r\mathbf{\Gamma}_{+1} & r\mathbf{\Gamma}_{+1} \\ (1-r)\mathbf{\Gamma}_{-1} & (1-r)\mathbf{\Gamma}_{-1} \end{bmatrix} \begin{bmatrix} \phi_{\mathbf{a}_{-1}}(\mathbf{X}) & \mathbf{0} \\ \mathbf{0} & \mathbf{A}_1 \\ \vdots & \vdots \\ \mathbf{0} & \mathbf{A}_n \end{bmatrix}^{\mathsf{T}}
$$

$$
+ \begin{bmatrix} r\nabla^2\Omega_{\mathbf{a}_{+1}} & r\mu_{+1,1}\mathbf{I} & \dots & r\mu_{+1,n}\mathbf{I} \\ (1-r)\mu_{-1,1}\mathbf{I} & (1-r)\nabla^2\Omega_{\mathbf{a}_{-1}} & \dots & \mathbf{0} \\ \vdots & \vdots & \ddots & \vdots \\ (1-r)\mu_{-1,n}\mathbf{I} & \mathbf{0} & \dots & (1-r)\nabla^2\Omega_{\mathbf{a}_{-1}} \end{bmatrix}. \tag{6}
$$

The eigenvalues of the inner matrix of the first summand in Equation 6 are $r\gamma_{+1,i} + (1-r)\gamma_{-1,i}$ and zero. Loss functions $\ell_v$ are convex in $y'$, that is, both second derivatives $\ell_v''(y', y)$ are non-negative for any $y'$ and consequently $r\gamma_{+1,i} + (1-r)\gamma_{-1,i} \geq 0$. Hence, the first summand of Jacobian $J_r(\mathbf{a})$ is positive semi-definite for any choice of $0 < r < 1$. Additionally, we can decompose the regularizers' Hessians as follows:

$$
\nabla^2\Omega_{\mathbf{a}_v} = \lambda_v\mathbf{I} + (\nabla^2\Omega_{\mathbf{a}_v} - \lambda_v\mathbf{I}), \tag{7}
$$

where $\lambda_v$ is the smallest eigenvalue of $\nabla^2\Omega_{\mathbf{a}_v}$. As the regularizers are strictly convex, $\lambda_v > 0$ and the second summand in Equation 7 is positive semi-definite. Hence, it suffices to show that matrix

$$
\begin{bmatrix} r\lambda_{+1}\mathbf{I} & r\mu_{+1,1}\mathbf{I} & \dots & r\mu_{+1,n}\mathbf{I} \\ (1-r)\mu_{-1,1}\mathbf{I} & (1-r)\lambda_{-1}\mathbf{I} & \dots & \mathbf{0} \\ \vdots & \vdots & \ddots & \vdots \\ (1-r)\mu_{-1,n}\mathbf{I} & \mathbf{0} & \dots & (1-r)\lambda_{-1}\mathbf{I} \end{bmatrix} \tag{8}
$$

is positive definite. We derive the eigenvalues of this matrix which assume only three different values; these are $(1-r)\lambda_{-1}$ and

$$
\frac{1}{2}\left(r\lambda_{+1} + (1-r)\lambda_{-1} \pm \sqrt{(r\lambda_{+1} - (1-r)\lambda_{-1})^2 + 4r(1-r)\mu_{+1}^{\mathsf{T}}\mu_{-1}}\right). \tag{9}
$$

Eigenvalue $(1-r)\lambda_{-1}$ is positive by definition. The others are positive if the value under the square root is non-negative and less than $(r\lambda_{+1} + (1-r)\lambda_{-1})^2$. The scalar product $b = \mu_{+1}^{\mathsf{T}}\mu_{-1}$ is non-positive as both loss functions $\ell_v(y', y)$ are monotonic in $y'$ with distinct monotonicity, *i.e.,* both derivatives have a different sign for any $y' \in \mathbb{R}$ and consequently $b \leq 0$. This implies that the value under the square root is less or equal to $(r\lambda_{+1} - (1-r)\lambda_{-1})^2 < (r\lambda_{+1} + (1-r)\lambda_{-1})^2$. In addition, $b$ is bounded from below as action spaces $A_v$, and therefore the value of $h_{\mathbf{a}_{+1}}(\phi_{\mathbf{a}_{-1}}(\mathbf{X})_i)$, is bounded. Let $\underline{b} = \inf_{\mathbf{a}\in A}\mu_{+1}^{\mathsf{T}}\mu_{-1}$ be such a lower bound with $-\infty < \underline{b} \leq 0$. We solve for $r$ such that the value under the square root in Equation 9 attains a non-negative value, that is,

$$
0 < r \leq \frac{(\lambda_{+1} + \lambda_{-1})\lambda_{-1} - 2\underline{b} - 2\sqrt{\underline{b}^2 - \lambda_{+1}\lambda_{-1}\underline{b}}}{(\lambda_{+1} + \lambda_{-1})^2 - 4\underline{b}} \tag{10}
$$

or alternatively

$$
\frac{(\lambda_{+1} + \lambda_{-1})\lambda_{-1} - 2\underline{b} + 2\sqrt{\underline{b}^2 - \lambda_{+1}\lambda_{-1}\underline{b}}}{(\lambda_{+1} + \lambda_{-1})^2 - 4\underline{b}} \leq r < 1. \tag{11}
$$

For any $\lambda_{+1}, \lambda_{-1} > 0$ there are values $r$ that satisfy Inequality 10 or 11 because, for any fixed $\underline{b} \leq 0$,

$$
0 < (\lambda_{+1} + \lambda_{-1})\lambda_{-1} - 2\underline{b} \pm 2\sqrt{\underline{b}^2 - \lambda_{+1}\lambda_{-1}\underline{b}} < (\lambda_{+1} + \lambda_{-1})^2 - 4\underline{b}. \tag{12}
$$

For such $r$ all eigenvalues in Equation 9 are strictly positive which completes the proof. $\square$

According to Theorem 1, a unique Nash equilibrium exists for suitable loss functions such as the squared hinge loss, logistic loss, etc. To find this equilibrium, we make use of the weighted Nikaido-Isoda function (Equation 13). Intuitively, $\Psi_{r_v}(\mathbf{a}, \mathbf{b})$ quantifies the weighted sum of the relative cost savings that the players can enjoy by changing from strategy $\mathbf{a}_v$ to strategy $\mathbf{b}_v$ while their opponent continues to play $\mathbf{a}_{-v}$. Equation 14 defines the value function $V_{r_v}(\mathbf{a})$ as the weighted sum of greatest

possible cost savings attainable by changing from $\mathbf{a}$ to any strategy unilaterally. By these definitions, $\mathbf{a}^*$ is a Nash equilibrium if, and only if, $V_{r_v}(\mathbf{a}^*)$ is a global minimum of the value function with $V_{r_v}(\mathbf{a}^*) = 0$ for any fixed weights $r_{+1} = r$ and $r_{-1} = 1 - r$, where $0 < r < 1$.

$$\Psi_{r_v}(\mathbf{a}, \mathbf{b}) = \sum_{v \in \{+1, -1\}} r_v(\theta_v(\mathbf{a}_v, \mathbf{a}_{-v}) - \theta_v(\mathbf{b}_v, \mathbf{a}_{-v})) \tag{13}$$

$$V_{r_v}(\mathbf{a}) = \max_{\mathbf{b} \in A} \Psi_{r_v}(\mathbf{a}, \mathbf{b}) \tag{14}$$

To find this global minimum of $V_{r_v}(\mathbf{a})$ we make use of Corollary 3.4 of [8]. The weights $r_v$ are fixed scaling factors of the players' objectives which do not affect the Nash equilibrium in Equation 2; however, these weights ensure the main condition of Corollary 3.4, that is, the positive definiteness of the Jacobian $J_r(\mathbf{a})$ in Equation 4. According to this corollary, vector $\mathbf{d} = \widehat{\mathbf{b}} - \mathbf{a}$ is a descent direction for the value function at any position $\mathbf{a}$, where $\widehat{\mathbf{b}}$ is the maximizing argument $\widehat{\mathbf{b}} = \arg\max_{\mathbf{b} \in A} \Psi_{r_v}(\mathbf{a}, \mathbf{b})$. In addition, the convexity of $A$ ensures that any point $\mathbf{a} + t\mathbf{d}$ with $t \in [0, 1]$ (*i.e.*, a point between $\mathbf{a}$ and $\widehat{\mathbf{b}}$) is a valid pair of actions.

---

**Algorithm 1** Nash Equilibrium of Games with Convex Loss Functions

---

**Require:** Cost functions $\theta_v$ as defined in Equation 1 and action spaces $A_v$.
 1: Select initial $\mathbf{a}^0 \in A_{+1} \times A_{-1}$, set $k := 0$, and choose $r$ that satisfies Inequality 10 or 11.
 2: **repeat**
 3:   Set $\mathbf{b}^k := \arg\max_{\mathbf{b} \in A_{+1} \times A_{-1}} \Psi_{r_v}(\mathbf{a}^k, \mathbf{b})$ where $\Psi_{r_v}$ is defined in Equation 13.
 4:   Set $\mathbf{d}^k := \mathbf{b}^k - \mathbf{a}^k$.
 5:   Find maximal step size $t^k \in \{2^{-l} : l \in \mathbb{N}\}$ with $V_{r_v}(\mathbf{a}^k + t^k\mathbf{d}^k) \leq V_{r_v}(\mathbf{a}^k) - \epsilon\|t^k\mathbf{d}^k\|^2$.
 6:   Set $\mathbf{a}^{k+1} := \mathbf{a}^k + t^k\mathbf{d}^k$ and $k := k + 1$.
 7: **until** $\|\mathbf{a}^k - \mathbf{a}^{k-1}\| \leq \epsilon$.

---

Algorithm 1 exploits these properties and finds the global minimum of $V_{r_v}$ and thereby the unique Nash equilibrium, under the preconditions of Theorem 1. Convergence follows from the fact that if in the $k$-th iteration $\mathbf{d}^k = \mathbf{0}$, then $\mathbf{a}^k$ is a Nash equilibrium which is unique according to Theorem 1. If $\mathbf{d}^k \neq \mathbf{0}$, then $\mathbf{d}^k$ is a descent direction of $V_{r_v}$ at position $\mathbf{a}^k$. Together with term $\epsilon\|t^k\mathbf{d}^k\|^2$, this ensures $V_{r_v}(\mathbf{a}^{k+1}) < V_{r_v}(\mathbf{a}^k)$, and as value function $V_{r_v}$ is bounded from below, Algorithm 1 converges to the global minimum of $V_{r_v}$. Note that $r$ only controls the convergence rate, but has no influence on the solution. Any value of $r$ that satisfies Inequality 10 or 11 ensures convergence.

## 4   Solution for Antagonistic Loss Functions

Algorithm 1 is guaranteed to identify the unique equilibrium if the loss functions are convex, twice differentiable, and of distinct monotonicities. We will now study the case in which the learner's *cost* function is continuous and convex, and the adversary's *loss* function is antagonistic to the learner's loss, that is, $\ell_{+1} = -\ell_{-1}$. We abstain from making assumptions about the adversary's regularizers. Because of the regularizers, the game is still not a zero-sum game. In this setting, a unique Nash equilibrium cannot be guaranteed to exist because the adversary's cost function is not necessarily strictly convex. However, an individual game *may* still possess a unique Nash equilibrium, and we can derive an algorithm that identifies it whenever it exists.

The symmetry of the loss functions simplifies the players' cost functions in Equation 1 to

$$\theta_{+1}(\mathbf{a}_{+1}, \mathbf{a}_{-1}) = \sum_{i=1}^{n} \ell_{+1}(h_{\mathbf{a}_{+1}}(\phi_{\mathbf{a}_{-1}}(\mathbf{X})_i), y_i) + \Omega_{\mathbf{a}_{+1}}, \tag{15}$$

$$\theta_{-1}(\mathbf{a}_{-1}, \mathbf{a}_{+1}) = -\sum_{i=1}^{n} \ell_{+1}(h_{\mathbf{a}_{+1}}(\phi_{\mathbf{a}_{-1}}(\mathbf{X})_i), y_i) + \Omega_{\mathbf{a}_{-1}}. \tag{16}$$

Even though the *loss* functions are antagonistic, the *cost* functions in Equations 15 and 16 are not, unless the player's regularizers are antagonistic as well. Hence, the game is not a zero-sum game. However, according to Theorem 2, if the game has a unique Nash equilibrium, then this equilibrium is a minimax solution of the zero-sum game defined by the joint cost function of Equation 17.

**Theorem 2.** *If the game with cost functions $\theta_{+1}$ and $\theta_{-1}$ defined in Equations 15 and 16 has a unique Nash equilibrium $\mathbf{a}^*$, then this equilibrium also satisfies $\mathbf{a}^* = \arg\min_{\mathbf{a}_{+1}} \max_{\mathbf{a}_{-1}} \theta_0(\mathbf{a}_{+1}, \mathbf{a}_{-1})$ where*

$$\theta_0(\mathbf{a}_{+1}, \mathbf{a}_{-1}) = \sum_{i=1}^{n} \ell_{+1}(h_{\mathbf{a}_{+1}}(\phi_{\mathbf{a}_{-1}}(\mathbf{X})_i), y_i) + \Omega_{\mathbf{a}_{+1}} - \Omega_{\mathbf{a}_{-1}}. \tag{17}$$

The proof can be found in the appendix. As a consequence of Theorem 2, we can identify the unique Nash equilibrium of the game with cost functions $\theta_{+1}$ and $\theta_{-1}$, if it exists, by finding the minimax solution of the game with joint cost function $\theta_0$. The minimax solution is given by

$$\mathbf{a}_{+1}^* = \arg\min_{\mathbf{a}_{+1} \in A_{+1}} \max_{\mathbf{a}_{-1} \in A_{-1}} \theta_0(\mathbf{a}_{+1}, \mathbf{a}_{-1}). \tag{18}$$

To solve this optimization problem, we define $\widehat{\theta}_0(\mathbf{a}_{+1}) = \theta_0(\mathbf{a}_{+1}, \widehat{\mathbf{a}}_{-1})$ to be the function of $\mathbf{a}_{+1}$ where $\widehat{\mathbf{a}}_{-1}$ is set to the value $\widehat{\mathbf{a}}_{-1} = \arg\max_{\mathbf{a}_{-1}} \theta_0(\mathbf{a}_{+1}, \mathbf{a}_{-1})$. Since cost function $\theta_0$ is continuous in its arguments, convex in $\mathbf{a}_{+1}$, and $A_{-1}$ is a compact set, Danskin's Theorem [9] implies that $\widehat{\theta}_0$ is convex in $\mathbf{a}_{+1}$ with gradient

$$\nabla\widehat{\theta}_0(\mathbf{a}_{+1}) = \nabla_{\mathbf{a}_{+1}}\theta_0(\mathbf{a}_{+1}, \widehat{\mathbf{a}}_{-1}). \tag{19}$$

The significance of Danskin's Theorem is that when calculating the gradient $\nabla_{\mathbf{a}_{+1}}\theta_0(\mathbf{a}_{+1}, \widehat{\mathbf{a}}_{-1})$ at position $\mathbf{a}_{+1}$, argument $\widehat{\mathbf{a}}_{-1}$ acts as a constant in the derivative instead of as a function of $\mathbf{a}_{+1}$. The convexity of $\widehat{\theta}_0(\mathbf{a}_{+1})$ suggests the gradient descent method implemented in Algorithm 2. It identifies the unique Nash equilibrium of a game with antagonistic loss functions, if it exists, by finding the minimax solution of the game with joint cost function $\theta_0$.

---

**Algorithm 2** Nash Equilibrium of Games with Antagonistic Loss Functions

---

**Require:** Joint cost function $\theta_0$ as defined in Equation 17 and action spaces $A_v$.
 1: Select initial $\mathbf{a}_{+1}^0 \in A_{+1}$ and set $k := 0$.
 2: **repeat**
 3:     Set $\mathbf{a}_{-1}^k := \arg\max_{\mathbf{a}_{-1} \in A_{-1}} \theta_0(\mathbf{a}_{+1}^k, \mathbf{a}_{-1})$.
 4:     Set $\mathbf{d}^k := -\nabla_{\mathbf{a}_{+1}^k} \theta_0(\mathbf{a}_{+1}^k, \mathbf{a}_{-1}^k)$.
 5:     Find maximal step size $t^k \in \{2^{-l} : l \in \mathbb{N}\}$ with

$$\theta_0(\mathbf{a}_{+1}^k + t^k \mathbf{d}^k, \mathbf{a}_{-1}^k) \leq \theta_0(\mathbf{a}_{+1}^k, \mathbf{a}_{-1}^k) - \epsilon \|t^k \mathbf{d}^k\|^2.$$

 6:     Set $\mathbf{a}_{+1}^{k+1} := \mathbf{a}_{+1}^k + t^k \mathbf{d}^k$ and $k := k + 1$.
 7:     Project $\mathbf{a}_{+1}^k$ to the admissible set $A_{+1}$, if necessary.
 8: **until** $\|\mathbf{a}_{+1}^k - \mathbf{a}_{+1}^{k-1}\| \leq \epsilon$

---

A minimax solution $\arg\min_{\mathbf{a}_{+1}} \max_{\mathbf{a}_{-1}} \theta_{+1}(\mathbf{a}_{+1}, \mathbf{a}_{-1})$ of the learner's cost function minimizes the learner's costs when playing against the most malicious opponent; for instance, Invar-SVM [4] finds such a solution. By contrast, the minimax solution $\arg\min_{\mathbf{a}_{+1}} \max_{\mathbf{a}_{-1}} \theta_0(\mathbf{a}_{+1}, \mathbf{a}_{-1})$ of the joint cost function as defined in Equation 17 constitutes a Nash equilibrium of the game with cost functions $\theta_{+1}$ and $\theta_{-1}$, defined in Equations 15 and 16. It minimizes the costs for each of two players that seek their personal advantage. Algorithmically, Invar-SVM and Algorithm 2 are very similar; the main difference lies in the optimization criteria and the resulting properties of the solution.

## 5 Experiments

We study the problem of email spam filtering where the learner tries to identify spam emails while the adversary conceals spam messages in order to penetrate the filter. Our goal is to explore the relative strengths and weaknesses of the proposed Nash models for antagonistic and non-antagonistic loss functions and existing baseline methods. We compare a *regular SVM*, *logistic regression*, *SVM with Invariances* (Invar-SVM, [4]), the *Nash equilibrium for antagonistic loss functions* found by identifying the minimax solution of the joint cost function (Minimax, Algorithm 2), and the *Nash equilibrium for convex loss functions* (Nash, Algorithm 1).

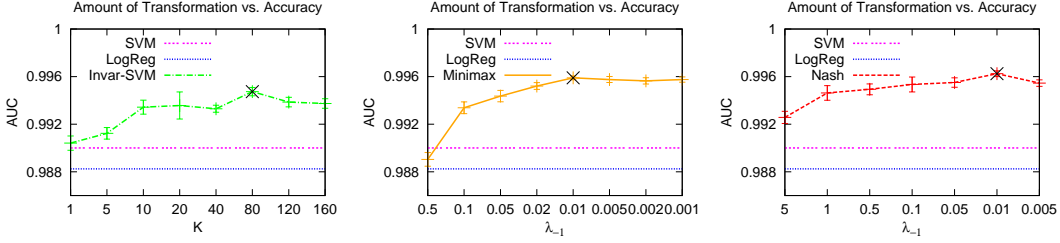

Figure 1: Adversary's regularization parameter and AUC on test data (private emails).

We use the logistic loss as the learner's loss function $\ell_{+1}(h(\mathbf{x}), y) = log(1 + e^{-yh(\mathbf{x})})$ for the Minimax and the Nash model. Consequently, the adversary's loss for the Minimax solution is the negative loss of the learner. In the Nash model, we choose $\ell_{-1}(h(\mathbf{x}), y) = log(1 + e^{yh(\mathbf{x})})$ which is a convex approximation of the adversary's zero-one loss, that is, correct predictions by the learner incur high costs for the adversary. We use the additive transformation model $\phi_{\mathbf{a}_{-1}}(\mathbf{X})_i = \mathbf{x}_i + \mathbf{a}_{-1,i}$ as defined in Section 2. For spam emails $\mathbf{x}_i$, we impose box constraints $-\frac{1}{2}\mathbf{x}_i \leq \mathbf{a}_{-1,i} \leq \frac{1}{2}\mathbf{x}_i$ on the adversary's parameters; for non-spam we set $\mathbf{a}_{-1,i} = \mathbf{0}$. That is, the spam sender can only transform spam emails. This model is equivalent to the component-wise scaling model [4] with scaling factors between $0.5$ and $1.5$, and ensures that the adversary's action space is nonempty, compact, and convex. We use $l_2$-norm regularizers for both players, that is, $\Omega_{\mathbf{a}_v} = \frac{\lambda_v}{2}\|\mathbf{a}_v\|_2^2$ where $\lambda_v$ is the regularization parameter of player $v$. For the Nash model we set $r$ to the mean of the interval defined by Inequality 11, where $\underline{b} = -\frac{n}{4}$ is a lower bound for the chosen logistic loss and regularization parameters $\lambda_v$ are identical to the smallest eigenvalues of $\nabla^2\Omega_{\mathbf{a}_v}$.

We use two email corpora: the first contains 65,000 publicly available emails received between 2000 and 2002 from the Enron corpus, the SpamAssassin corpus, Bruce Guenter's spam trap, and several mailing lists. The second contains 40,000 private emails received between 2000 and 2007. All emails are binary word vectors of dimensionality 329,518 and 160,981, respectively. The emails are sorted chronologically and tagged with label, date, and size. The preprocessed corpora are available from the authors. We cannot use a standard TREC corpus because there the delivery dates of the spam messages have been fabricated, and our experiments require the correct chronological order.

Our evaluation protocol is as follows. We use the 6,000 oldest instances as training portion and set the remaining emails aside as test instances. We use the area under the ROC curve as a fair evaluation metric that is adequate for the application; error bars indicate the standard error. We train all methods 20 times for the first experiment and 50 times for the following experiments on a subset of 200 messages drawn at random from the training portion and average the AUC values on the test set. In order to tune both players' regularization parameters, we conduct a grid search maximizing the AUC for 5-fold cross validation on the training portion.

In the first experiment, we explore the impact of the regularization parameter of the transformation model, *i.e.,* $\lambda_{-1}$ for our models and $K$ – the maximal number of alterable attributes – for Invar-SVM. Figure 1 shows the averaged AUC value on the private corpus' test portion. The crosses indicate the parameter values found by the grid search with cross validation on the training data.

In the next experiment, we evaluate all methods *into the future* by processing the test set in chronological order. Figure 2 shows that Invar-SVM, Minimax, and the Nash solution outperform the regular SVM and logistic regression significantly. For the public data set, Minimax performs slightly better than Nash; for the private corpus, there is no significant difference between the solutions of Minimax and Nash. For both data sets, the $l_2$-regularization gives Minimax and Nash an advantage over Invar-SVM. Recall that Minimax refers to the Nash equilibrium for antagonistic loss functions found by solving the minimax problem for the joint cost function (Algorithm 2). In this setting, loss functions – but not cost functions – are antagonistic; hence, Nash cannot gain an advantage over Minimax. Figure 2 (right hand side) shows the execution time of all methods. Regular SVM and logistic regression are faster than the game models; the game models behave comparably.

Finally, we explore a setting with non-antagonistic loss. We weight the loss functions with player- and instance specific factors $c_{v,i}$, that is, $\ell_v^c(h_{\mathbf{a}_{+1}}(\phi_{\mathbf{a}_{-1}}(\mathbf{X})_i), y_i) = c_{v,i}\ell_v(h_{\mathbf{a}_{+1}}(\phi_{\mathbf{a}_{-1}}(\mathbf{X})_i), y_i)$.

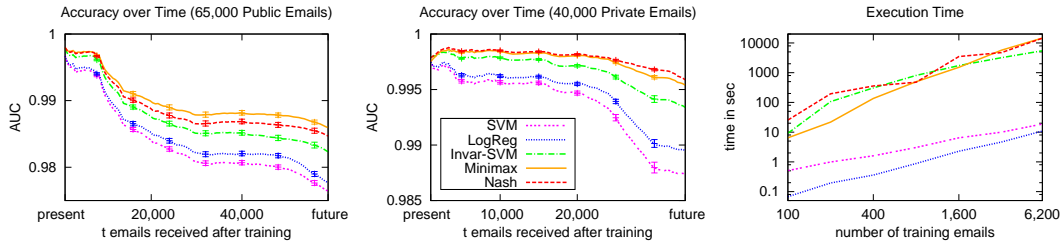

Figure 2: Left, center: AUC evaluated into the future after training on past. Right: execution time.

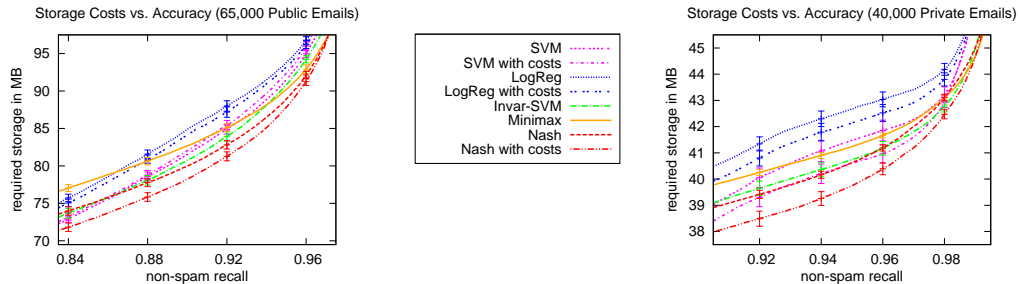

Figure 3: Average storage costs versus non-spam recall.

Our model reflects that an email service provider may delete detected spam emails after a latency period whereas other emails incur storage costs $c_{+1,i}$ proportional to their file size. The spam sender's costs are $c_{-1,i} = 1$ for all spam instances and $c_{-1,i} = 0$ for all non-spam instances. The classifier threshold balances a trade-off between non-spam recall (fraction of legitimate emails delivered) and storage costs. For a threshold of $-\infty$, storage costs and non-spam recall are zero for all decision functions. Likewise, a threshold of $\infty$ gives a recall of 1, but all emails have to be stored. Figure 3 shows this trade-off for all methods. The Nash prediction model behaves most favorably: it outperforms all reference methods for almost all threshold values, often by several standard errors. Invar-SVM and Minimax cannot reflect differing costs for learner and adversary in their optimization criteria and therefore perform worse. Logistic regression and the SVM with costs perform better than their counterparts without costs, but worse than the Nash model.

# 6 Conclusion

We studied games in which each player's cost function consists of a data-dependent loss and a regularizer. A learner produces a linear model while an adversary chooses a transformation matrix to be added to the data matrix. Our main result regards regularized non-antagonistic loss functions that are convex, twice differentiable, and have distinct monotonicity. In this case, a unique Nash equilibrium exists. It minimizes the costs of each of two players that aim for their highest personal benefit. We derive an algorithm that identifies the equilibrium under these conditions. For the case of antagonistic loss functions with arbitrary regularizers a unique Nash equilibrium may or may not exist. We derive an algorithm that finds the unique Nash equilibrium, if it exists, by solving a minimax problem on a newly derived joint cost function.

We evaluate spam filters derived from the different optimization problems on chronologically ordered future emails. We observe that game models outperform the reference methods. In a setting with player- and instance-specific costs, the Nash model for non-antagonistic loss functions excels because this setting is poorly modeled with antagonistic loss functions.

### Acknowledgments

We gratefully acknowledge support from STRATO AG.

# References

[1] Gert R. G. Lanckriet, Laurent El Ghaoui, Chiranjib Bhattacharyya, and Michael I. Jordan. A robust minimax approach to classification. *Journal of Machine Learning Research*, 3:555–582, 2002.

[2] Laurent El Ghaoui, Gert R. G. Lanckriet, and Georges Natsoulis. Robust classification with interval data. Technical Report UCB/CSD-03-1279, EECS Department, University of California, Berkeley, 2003.

[3] Amir Globerson and Sam T. Roweis. Nightmare at test time: robust learning by feature deletion. In *Proceedings of the International Conference on Machine Learning*, 2006.

[4] Choon Hui Teo, Amir Globerson, Sam T. Roweis, and Alex J. Smola. Convex learning with invariances. In *Advances in Neural Information Processing Systems*, 2008.

[5] Amir Globerson, Choon Hui Teo, Alex J. Smola, and Sam T. Roweis. *Dataset Shift in Machine Learning*, chapter An adversarial view of covariate shift and a minimax approach, pages 179–198. MIT Press, 2009.

[6] Tamer Basar and Geert J. Olsder. *Dynamic Noncooperative Game Theory*. Society for Industrial and Applied Mathematics, 1999.

[7] J. B. Rosen. Existence and uniqueness of equilibrium points for concave n-person games. *Econometrica*, 33(3):520–534, 1965.

[8] Anna von Heusinger and Christian Kanzow. Relaxation methods for generalized Nash equilibrium problems with inexact line search. *Journal of Optimization Theory and Applications*, 143(1):159–183, 2009.

[9] John M. Danskin. The theory of max-min, with applications. *SIAM Journal on Applied Mathematics*, 14(4):641–664, 1966.

